# Streaming $k$-means approximation

**Nir Ailon**
Google Research
nailon@google.com

**Ragesh Jaiswal**[*]
Columbia University
rjaiswal@gmail.com

**Claire Monteleoni**[†]
Columbia University
cmontel@ccls.columbia.edu

## Abstract

We provide a clustering algorithm that approximately optimizes the $k$-means objective, in the one-pass streaming setting. We make no assumptions about the data, and our algorithm is very light-weight in terms of memory, and computation. This setting is applicable to unsupervised learning on massive data sets, or resource-constrained devices. The two main ingredients of our theoretical work are: a derivation of an extremely simple pseudo-approximation batch algorithm for $k$-means (based on the recent $k$-means++), in which the algorithm is allowed to output more than $k$ centers, and a streaming clustering algorithm in which batch clustering algorithms are performed on small inputs (fitting in memory) and combined in a hierarchical manner. Empirical evaluations on real and simulated data reveal the practical utility of our method.

## 1 Introduction

As commercial, social, and scientific data sources continue to grow at an unprecedented rate, it is increasingly important that algorithms to process and analyze this data operate in online, or one-pass streaming settings. The goal is to design light-weight algorithms that make only one pass over the data. Clustering techniques are widely used in machine learning applications, as a way to summarize large quantities of high-dimensional data, by partitioning them into "clusters" that are useful for the specific application. The problem with many heuristics designed to implement some notion of clustering is that their outputs can be hard to evaluate. Approximation guarantees, with respect to some reasonable objective, are therefore useful. The $k$-means objective is a simple, intuitive, and widely-cited clustering objective for data in Euclidean space. However, although many clustering algorithms have been designed with the $k$-means objective in mind, very few have approximation guarantees with respect to this objective.

In this work, we give a one-pass streaming algorithm for the $k$-means problem. We are not aware of previous approximation guarantees with respect to the $k$-means objective that have been shown for simple clustering algorithms that operate in either online or streaming settings. We extend work of Arthur and Vassilvitskii [AV07] to provide a bi-criterion approximation algorithm for $k$-means, in the batch setting. They define a seeding procedure which chooses a subset of $k$ points from a batch of points, and they show that this subset gives an expected $O(\log(k))$-approximation to the $k$-means objective. This seeding procedure is followed by Lloyd's algorithm[1] which works very well in practice with the seeding. The combined algorithm is called $k$-means++, and is an $O(\log(k))$-approximation algorithm, in expectation.[2] We modify $k$-means++ to obtain a new algorithm, $k$-means#, which chooses a subset of $O(k \log(k))$ points, and we show that the chosen subset of

---

[*]Department of Computer Science. Research supported by DARPA award HR0011-08-1-0069.

[†]Center for Computational Learning Systems

[1]Lloyd's algorithm is popularly known as the $k$-means algorithm

[2]Since the approximation guarantee is proven based on the seeding procedure alone, for the purposes of this exposition we denote the seeding procedure as $k$-means++.

points gives a constant approximation to the $k$-means objective. Apart from giving us a bi-criterion approximation algorithm, our modified seeding procedure is very simple to analyze.

[GMMM+03] defines a divide-and-conquer strategy to combine multiple bi-criterion approximation algorithms for the $k$-medoid problem to yield a one-pass streaming approximation algorithm for $k$-median. We extend their analysis to the $k$-means problem and then use $k$-means++ and $k$-means# in the divide-and-conquer strategy, yielding an extremely efficient single pass streaming algorithm with an $O(c^\alpha \log(k))$-approximation guarantee, where $\alpha \approx \log n / \log M$, $n$ is the number of input points in the stream and $M$ is the amount of work memory available to the algorithm. Empirical evaluations, on simulated and real data, demonstrate the practical utility of our techniques.

## 1.1 Related work

There is much literature on both clustering algorithms [Gon85, Ind99, VW02, GMMM+03, KMNP+04, ORSS06, AV07, CR08, BBG09, AL09], and streaming algorithms [Ind99, GMMM+03, M05, McG07].[3] There has also been work on combining these settings: designing clustering algorithms that operate in the streaming setting [Ind99, GMMM+03, CCP03]. Our work is inspired by that of Arthur and Vassilvitskii [AV07], and Guha *et al.* [GMMM+03], which we mentioned above and will discuss in further detail. $k$-means++, the seeding procedure in [AV07], had previously been analyzed by [ORSS06], under special assumptions on the input data.

In order to be useful in machine learning applications, we are concerned with designing algorithms that are extremely light-weight and practical. $k$-means++ is efficient, very simple, and performs well in practice. There do exist constant approximations to the $k$-means objective, in the non-streaming setting, such as a local search technique due to [KMNP+04].[4] A number of works [LV92, CG99, Ind99, CMTS02, AGKM+04] give constant approximation algorithms for the related $k$-median problem in which the objective is to minimize the sum of distances of the points to their nearest centers (rather than the square of the distances as in $k$-means), and the centers must be a subset of the input points. It is popularly believed that most of these algorithms can be extended to work for the $k$-means problem without too much degradation of the approximation, however there is no formal evidence for this yet. Moreover, the running times of most of these algorithms depend worse than linearly on the parameters $(n, k, \text{and } d)$ which makes these algorithms less useful in practice. As future work, we propose analyzing variants of these algorithms in our streaming clustering algorithm, with the goal of yielding a streaming clustering algorithm with a *constant* approximation to the $k$-means objective.

Finally, it is important to make a distinction from some lines of clustering research which involve assumptions on the data to be clustered. Common assumptions include i.i.d. data, *e.g.* [BL08], and data that admits a clustering with well separated means *e.g.* in [VW02, ORSS06, CR08]. Recent work [BBG09] assumes a "target" clustering for the specific application and data set, that is close to any constant approximation of the clustering objective. In contrast, we prove approximation guarantees with respect to the optimal $k$-means clustering, with no assumptions on the input data.[5] As in [AV07], our probabilistic guarantees are only with respect to randomness in the algorithm.

### 1.1.1 Preliminaries

The $k$-means clustering problem is defined as follows: Given $n$ points $\mathcal{X} \subset \mathbb{R}^d$ and a weight function $w : \mathcal{X} \to \mathbb{R}$, the goal is to find a subset $\mathcal{C} \subseteq \mathbb{R}^d$, $|\mathcal{C}| = k$ such that the following quantity is minimized:[6] $\phi_{\mathcal{C}} = \sum_{x \in \mathcal{X}} w(x) \cdot D(x, \mathcal{C})^2$, where $D(x, \mathcal{C})$ denotes the $\ell_2$ distance of $x$ to the nearest point in $\mathcal{C}$. When the subset $\mathcal{C}$ is clear from the context, we denote this distance by $D(x)$. Also, for two points $x, y$, $D(x, y)$ denotes the $\ell_2$ distance between $x$ and $y$. The subset $\mathcal{C}$ is alternatively called a clustering of $\mathcal{X}$ and $\phi_{\mathcal{C}}$ is called the potential function corresponding to the clustering. We will use the term "center" to refer to any $c \in \mathcal{C}$.

**Definition 1.1** (Competitive ratio, $b$-approximation). Given an algorithm $B$ for the $k$-means problems, let $\phi_{\mathcal{C}}$ be the potential of the clustering $\mathcal{C}$ returned by $B$ (on some input set which is implicit) and let $\phi_{\mathcal{C}_{OPT}}$ denote the potential of the optimal clustering $\mathcal{C}_{OPT}$. Then the competitive ratio is defined to be the worst case ratio $\frac{\phi_{\mathcal{C}}}{\phi_{\mathcal{C}_{OPT}}}$. The algorithm $B$ is said to be $b$-approximation algorithm if $\frac{\phi_{\mathcal{C}}}{\phi_{\mathcal{C}_{OPT}}} \leq b$.

The previous definition might be too strong for an approximation algorithm for some purposes. For example, the clustering algorithm performs poorly when it is constrained to output $k$ centers but it might become competitive when it is allowed to output more centers.

**Definition 1.2** ($(a, b)$-approximation). We call an algorithm $B$, $(a, b)$-approximation for the $k$-means problem if it outputs a clustering $\mathcal{C}$ with $ak$ centers with potential $\phi_{\mathcal{C}}$ such that $\frac{\phi_{\mathcal{C}}}{\phi_{\mathcal{C}_{OPT}}} \leq b$ in the worst case. Where $a > 1, b > 1$.

Note that for simplicity, we measure the memory in terms of the *words* which essentially means that we assume a point in $\mathbb{R}^d$ can be stored in $O(1)$ space.

## 2    $k$-means#: The advantages of careful and liberal seeding

The $k$-means++ algorithm is an expected $\Theta(\log k)$-approximation algorithm. In this section, we extend the ideas in [AV07] to get an $(O(\log k), O(1))$-approximation algorithm. Here is the $k$-means++ algorithm:

---

1. Choose an initial center $c_1$ uniformly at random from $\mathcal{X}$.
2. Repeat $(k-1)$ times:
3.     Choose the next center $c_i$, selecting $c_i = x' \in \mathcal{X}$ with probability $\frac{D(x')^2}{\sum_{x \in \mathcal{X}} D(x)^2}$.
   *(here $D(.)$ denotes the distances w.r.t. to the subset of points chosen in the previous rounds)*

---

**Algorithm 1:** $k$-means++

In the original definition of $k$-means++ in [AV07], the above algorithm is followed by Lloyd's algorithm. The above algorithm is used as a seeding step for Lloyd's algorithm which is known to give the best results in practice. On the other hand, the theoretical guarantee of the $k$-means++ comes from analyzing this seeding step and not Lloyd's algorithm. So, for our analysis we focus on this seeding step. The running time of the algorithm is $O(nkd)$.

In the above algorithm $\mathcal{X}$ denotes the set of given points and for any point $x$, $D(x)$ denotes the distance of this point from the nearest center among the centers chosen in the previous rounds. To get an $(O(\log k), O(1))$-approximation algorithm, we make a simple change to the above algorithm. We first set up the tools for analysis. These are the basic lemmas from [AV07]. We will need the following definition first:

**Definition 2.1** (Potential w.r.t. a set). Given a clustering $\mathcal{C}$, its potential with respect to some set $A$ is denoted by $\phi_{\mathcal{C}}(A)$ and is defined as $\phi_{\mathcal{C}}(A) = \sum_{x \in A} D(x)^2$, where $D(x)$ is the distance of the point $x$ from the nearest point in $\mathcal{C}$.

**Lemma 2.2** ([AV07], Lemma 3.1). *Let $A$ be an arbitrary cluster in $\mathcal{C}_{OPT}$, and let $\mathcal{C}$ be the clustering with just one center, chosen uniformly at random from $A$. Then $\mathbf{Exp}[\phi_{\mathcal{C}}(A)] = 2 \cdot \phi_{\mathcal{C}_{OPT}}(A)$.*

**Corollary 2.3.** *Let $A$ be an arbitrary cluster in $\mathcal{C}_{OPT}$, and let $\mathcal{C}$ be the clustering with just one center, which is chosen uniformly at random from $A$. Then, $\mathbf{Pr}[\phi_{\mathcal{C}}(A) < 8\phi_{\mathcal{C}_{OPT}}(A)] \geq 3/4$*

*Proof.* The proof follows from Markov's inequality. $\qquad\qquad\qquad\qquad\qquad\qquad\qquad\square$

**Lemma 2.4** ([AV07], Lemma 3.2). *Let $A$ be an arbitrary cluster in $\mathcal{C}_{OPT}$, and let $\mathcal{C}$ be an arbitrary clustering. If we add a random center to $\mathcal{C}$ from $A$, chosen with $D^2$ weighting to get $\mathcal{C}'$, then $\mathbf{Exp}[\phi_{\mathcal{C}'}(A)] \leq 8 \cdot \phi_{\mathcal{C}_{OPT}}(A)$.*

**Corollary 2.5.** *Let $A$ be an arbitrary cluster in $\mathcal{C}_{OPT}$, and let $\mathcal{C}$ be an arbitrary clustering. If we add a random center to $\mathcal{C}$ from $A$, chosen with $D^2$ weighting to get $\mathcal{C}'$, then $\mathbf{Pr}[\phi_{\mathcal{C}'}(A) < 32 \cdot \phi_{\mathcal{C}_{OPT}}(A)] \geq 3/4$.*

We will use $k$-means++ and the above two lemmas to obtain a $(O(\log k), O(1))$-approximation algorithm for the $k$-means problem. Consider the following algorithm:

---

1. Choose $3 \cdot \log k$ centers independently and uniformly at random from $\mathcal{X}$.
2. Repeat $(k-1)$ times.
3.      Choose $3 \cdot \log k$ centers independently and with probability $\frac{D(x')^2}{\sum_{x \in \mathcal{X}} D(x)^2}$.

    *(here $D(.)$ denotes the distances w.r.t. to the subset of points chosen in the previous rounds)*

---

**Algorithm 2:** $k$-means#

Note that the algorithm is almost the same as the $k$-means++ algorithm except that in each round of choosing centers, we pick $O(\log k)$ centers rather than a single center. The running time of the above algorithm is clearly $O(ndk \log k)$.

Let $\mathcal{A} = \{A_1, ..., A_k\}$ denote the set of clusters in the optimal clustering $\mathcal{C}_{OPT}$. Let $\mathcal{C}^i$ denote the clustering after $i^{th}$ round of choosing centers. Let $\mathcal{A}_c^i$ denote the subset of clusters $\in \mathcal{A}$ such that

$$\forall A \in \mathcal{A}_c^i, \ \phi_{\mathcal{C}^i}(A) \leq 32 \cdot \phi_{\mathcal{C}_{OPT}}(A).$$

We call this subset of clusters, the "covered" clusters. Let $\mathcal{A}_u^i = \mathcal{A} \setminus \mathcal{A}_c^i$ be the subset of "uncovered" clusters. The following simple lemma shows that with constant probability step (1) of $k$-means# picks a center such that at least one of the clusters gets covered, or in other words, $|\mathcal{A}_c^1| \geq 1$. Let us call this event $E$.

**Lemma 2.6.** $\mathbf{Pr}[E] \geq (1 - 1/k)$.

*Proof.* The proof easily follows from Corollary 2.3.             $\square$

Let $\mathcal{X}_c^i = \cup_{A \in \mathcal{A}_c^i} A$ and let $\mathcal{X}_u^i = \mathcal{X} \setminus \mathcal{X}_c^i$. Now after the $i^{th}$ round, either $\phi_{\mathcal{C}^i}(\mathcal{X}_c^i) \leq \phi_{\mathcal{C}^i}(\mathcal{X}_u^i)$ or otherwise. In the former case, using Corollary 2.5, we show that the probability of covering an uncovered cluster in the $(i+1)^{th}$ round is large. In the latter case, we will show that the current set of centers is already competitive with constant approximation ratio. Let us start with the latter case.

**Lemma 2.7.** *If event $E$ occurs ( $|\mathcal{A}_c^1| \geq 1$) and for any $i > 1$, $\phi_{\mathcal{C}^i}(\mathcal{X}_c^i) > \phi_{\mathcal{C}^i}(\mathcal{X}_u^i)$, then $\phi_{\mathcal{C}^i} \leq 64\phi_{\mathcal{C}_{OPT}}$.*

*Proof.* We get the main result using the following sequence of inequalities: $\phi_{\mathcal{C}^i} = \phi_{\mathcal{C}^i}(\mathcal{X}_c^i) + \phi_{\mathcal{C}^i}(\mathcal{X}_u^i) \leq \phi_{\mathcal{C}^i}(\mathcal{X}_c^i) + \phi_{\mathcal{C}^i}(\mathcal{X}_c^i) \leq 2 \cdot 32 \cdot \phi_{\mathcal{C}_{OPT}}(\mathcal{X}_c^i) \leq 64 \ \phi_{\mathcal{C}_{OPT}}$ (using the definition of $\mathcal{X}_c^i$).    $\square$

**Lemma 2.8.** *If for any $i \geq 1$, $\phi_{\mathcal{C}^i}(\mathcal{X}_c^i) \leq \phi_{\mathcal{C}^i}(\mathcal{X}_u^i)$, then $\mathbf{Pr}[|\mathcal{A}_c^{i+1}| \geq |\mathcal{A}_c^i| + 1] \geq (1 - 1/k)$.*

*Proof.* Note that in the $(i+1)^{th}$ round, the probability that a center is chosen from a cluster $\notin \mathcal{A}_c^i$ is at least $\frac{\phi_{\mathcal{C}^i}(\mathcal{X}_u^i)}{\phi_{\mathcal{C}^i}(\mathcal{X}_u^i) + \phi_{\mathcal{C}^i}(\mathcal{X}_c^i)} \geq 1/2$. Conditioned on this event, with probability at least $3/4$ any of the centers $x$ chosen in round $(i+1)$ satisfies $\phi_{\mathcal{C}^i \cup x}(A) \leq 32 \cdot \phi_{\mathcal{C}_{OPT}}(A)$ for some uncovered cluster $A \in \mathcal{A}_u^i$. This means that with probability at least $3/8$ any of the chosen centers $x$ in round $(i+1)$ satisfies $\phi_{\mathcal{C}^i \cup x}(A) \leq 32 \cdot \phi_{\mathcal{C}_{OPT}}(A)$ for some uncovered cluster $A \in \mathcal{A}_u^i$. This further implies that with probability at least $(1 - 1/k)$ at least one of the chosen centers $x$ in round $(i+1)$ satisfies $\phi_{\mathcal{C}^i \cup x}(A) \leq 32 \cdot \phi_{\mathcal{C}_{OPT}}(A)$ for some uncovered cluster $A \in \mathcal{A}_u^i$.    $\square$

We use the above two lemmas to prove our main theorem.

**Theorem 2.9.** *$k$-means# is a $(O(\log k), O(1))$-approximation algorithm.*

*Proof.* From Lemma 2.6 we know that event $E$ (i.e., $|\mathcal{A}_c^i| \geq 1$) occurs. Given this, suppose for any $i > 1$, after the $i^{th}$ round $\phi_{\mathcal{C}^i}(\mathcal{X}_c) > \phi_{\mathcal{C}^i}(\mathcal{X}_u)$. Then from Lemma 2.7 we have $\phi_{\mathcal{C}} \leq \phi_{\mathcal{C}^i} \leq 64\phi_{\mathcal{C}_{OPT}}$. If no such $i$ exist, then from Lemma 2.8 we get that the probability that there exists a cluster $A \in \mathcal{A}$ such that $A$ is not covered even after $k$ rounds(i.e., end of the algorithm) is at most: $1 - (1 - 1/k)^k \leq 3/4$. So with probability at least $1/4$, the algorithm covers all the clusters in $\mathcal{A}$. In this case from Lemma 2.8, we have $\phi_{\mathcal{C}} = \phi_{\mathcal{C}^k} \leq 32 \cdot \phi_{\mathcal{C}_{OPT}}$.    $\square$

We have shown that $k$-means# is a randomized algorithm for clustering which with probability at least $1/4$ gives a clustering with competitive ratio 64.

# 3 A single pass streaming algorithm for $k$-means

In this section, we will provide a single pass streaming algorithm. The basic ingredients for the algorithm is a divide and conquer strategy defined by [GMMM+03] which uses bi-criterion approximation algorithms in the batch setting. We will use $k$-means++ which is a $(1, O(\log k))$-approximation algorithm and $k$-means# which is a $(O(\log k), O(1))$-approximation algorithm, to construct a single pass streaming $O(\log k)$-approximation algorithm for $k$-means problem. In the next subsection, we develop some of the tools needed for the above.

## 3.1 A streaming (a,b)-approximation for $k$-means

We will show that a simple streaming divide-and-conquer scheme, analyzed by [GMMM+03] with respect to the $k$-medoid objective, can be used to approximate the $k$-means objective. First we present the scheme due to [GMMM+03], where in this case we use $k$-means-approximating algorithms as input.

---

**Inputs:** (a) Point set $S \subset R^d$. Let $n = |S|$.
         (b) Number of desired clusters, $k \in N$.
         (c) $A$, an $(a, b)$-approximation algorithm to the $k$-means objective.
         (d) $A'$, an $(a', b')$-approximation algorithm to the $k$-means objective.

1. Divide $S$ into groups $S_1, S_2, \ldots, S_\ell$
2. For each $i \in \{1, 2, \ldots, \ell\}$
3.     Run A on $S_i$ to get $\leq ak$ centers $T_i = \{t_{i1}, t_{i2}, \ldots\}$
4.     Denote the induced clusters of $S_i$ as $S_{i1} \cup S_{i2} \cup \cdots$
5. $S_w \leftarrow T_1 \cup T_2 \cup \cdots \cup T_\ell$, with weights $w(t_{ij}) \leftarrow |S_{ij}|$
6. Run $A'$ on $S_w$ to get $\leq a'k$ centers $T$
7. Return $T$

**Algorithm 3:** [GMMM+03] Streaming divide-and-conquer clustering

---

First note that when every batch $S_i$ has size $\sqrt{nk}$, this algorithm takes one pass, and $O(a\sqrt{nk})$ memory. Now we will give an approximation guarantee.

**Theorem 3.1.** *The algorithm above outputs a clustering that is an $(a', 2b + 4b'(b + 1))$-approximation to the $k$-means objective.*

The $a'$ approximation of the desired number of centers follows directly from the approximation property of $A'$, with respect to the number of centers, since $A'$ is the last algorithm to be run. It remains to show the approximation of the $k$-means objective. The proof, which appears in the Appendix, involves extending the analysis of [GMMM+03], to the case of the $k$-means objective. Using the exposition in Dasgupta's lecture notes [Das08], of the proof due to [GMMM+03], our extension is straightforward, and differs in the following ways from the $k$-medoid analysis.

1. The $k$-means objective involves squared distance (as opposed to $k$-medoid in which the distance is not squared), so the triangle inequality cannot be invoked directly. We replace it with an application of the triangle inequality, followed by $(a+b)^2 \leq 2a^2 + 2b^2$, everywhere it occurs, introducing several factors of 2.

2. Cluster centers are chosen from $\mathbb{R}^d$, for the $k$-means problem, so in various parts of the proof we save an approximation a factor of 2 from the $k$-medoid problem, in which cluster centers must be chosen from the input data.

## 3.2 Using $k$-means++ and $k$-means# in the divide-and-conquer strategy

In the previous subsection, we saw how a $(a, b)$-approximation algorithm $A$ and an $(a', b')$-approximation algorithm $A'$ can be used to get a single pass $(a', 2b + 4b'(b + 1))$-approximation streaming algorithm. We now have two randomized algorithms, $k$-means# which with probability at least $1/4$ is a $(3 \log k, 64)$-approximation algorithm and $k$-means++ which is a $(1, O(\log k))$-approximation algorithm (the approximation factor being in expectation). We can now use these two algorithms in the divide-and-conquer strategy to obtain a single pass streaming algorithm.

We use the following as algorithms as $A$ and $A'$ in the divide-and-conquer strategy (3):

**A**: "Run $k$-means# on the data $3 \log n$ times independently, and pick the clustering with the smallest cost."
**A'**: "Run $k$-means++"

**Weighted versus non-weighted.** Note that $k$-means and $k$-means# are approximation algorithms for the non-weighted case (*i.e.* $w(x) = 1$ for all points $x$). On the other hand, in the divide-and-conquer strategy we need the algorithm $A'$, to work for the weighted case where the weights are integers. Note that both $k$-means and $k$-means# can be easily generalized for the weighted case when the weights are integers. Both algorithms compute probabilities based on the cost with respect to the current clustering. This cost can be computed by taking into account the weights. For the analysis, we can assume points with multiplicities equal to the integer weight of the point. The memory required remains logarithmic in the input size, including the storing the weights.

**Analysis.** With probability at least $\left(1 - (3/4)^{3 \log n}\right) \geq \left(1 - \frac{1}{n}\right)$, algorithm $A$ is a $(3 \log k, 64)$-approximation algorithm. Moreover, the space requirement remains logarithmic in the input size. In step (3) of Algorithm 3 we run $A$ on batches of data. Since each batch is of size $\sqrt{nk}$ the number of batches is $\sqrt{n/k}$, the probability that $A$ is a $(3 \log k, 64)$-approximation algorithm for all of these batches is at least $\left(1 - \frac{1}{n}\right)^{\sqrt{n/k}} \geq 1/2$. Conditioned on this event, the divide-and-conquer strategy gives a $O(\log k)$-approximation algorithm. The memory required is $O(\log(k) \cdot \sqrt{nk})$ times the logarithm of the input size. Moreover, the algorithm has running time $O(dnk \log n \log k)$.

### 3.3 Improved memory-approximation tradeoffs

We saw in the last section how to obtain a single-pass $(a', cbb')$-approximation for $k$-means using first an $(a, b)$-approximation on input blocks and then an $(a', b')$-approximation on the union of the output center sets, where $c$ is some global constant. The optimal memory required for this scheme was $O(a\sqrt{nk})$. This immediately implies a tradeoff between the memory requirements (growing like $a$), the number of centers outputted (which is $a'k$) and the approximation to the potential (which is $cbb'$) with respect to the optimal solution using $k$ centers. A more subtle tradeoff is possible by a recursive application of the technique in multiple levels. Indeed, the $(a, b)$-approximation could be broken up in turn into two levels, and so on. This idea was used in [GMMM+03]. Here we make a more precise account of the tradeoff between the different parameters.

Assume we have subroutines for performing $(a_i, b_i)$-approximation for $k$-means in batch mode, for $i = 1, \ldots r$ (we will choose $a_1, \ldots, a_r, b_1, \ldots, b_r$ later). We will hold $r$ buffers $B_1, \ldots, B_r$ as work areas, where the size of buffer $B_i$ is $M_i$. In the topmost level, we will divide the input into equal blocks of size $M_1$, and run our $(a_1, b_1)$-approximation algorithm on each block. Buffer $B_1$ will be repeatedly reused for this task, and after each application of the approximation algorithm, the outputted set of (at most) $ka_1$ centers will be added to $B_2$. When $B_2$ is filled, we will run the $(a_2, b_2)$-approximation algorithm on the data and add the $ka_2$ outputted centers to $B_3$. This will continue until buffer $B_r$ fills, and the $(a_r, b_r)$-approximation algorithm outputs the final $a_r k$ centers. Let $t_i$ denote the number of times the $i$'th level algorithm is executed. Clearly we have $t_i k a_i = M_{i+1} t_{i+1}$ for $i = 1, \ldots, r - 1$. For the last stage we have $t_r = 1$, which means that $t_{r-1} = M_r/ka_{r-1}$, $t_{r-2} = M_{r-1}M_r/k^2 a_{r-2}a_{r-1}$ and generally $t_i = M_{i+1} \cdots M_r/k^{r-i}a_i \cdots a_{r-1}$.[7] But we must also have $t_1 = n/M_1$, implying $n = \frac{M_1 \cdots M_r}{k^{r-1}a_1 \cdots a_{r-1}}$. In order to minimize the total memory $\sum M_i$ under the last constraint, using standard arguments in multivariate analysis we must have $M_1 = \cdots = M_r$, or in other words $M_i = \left(nk^{r-1}a_1 \cdots a_{r-1}\right)^{1/r} \leq n^{1/r}k(a_1 \cdots a_{r-1})^{1/r}$ for all $i$. The resulting one-pass algorithm will have an approximation guarantee of $(a_r, c^{r-1}b_1 \cdots b_r)$ (using a straightforward extension of the result in the previous section) and memory requirement of at most $rn^{1/r}k(a_1 \cdots a_{r-1})^{1/r}$.

Assume now that we are in the realistic setting in which the available memory is of fixed size $M \geq k$. We will choose $r$ (below), and for each $i = 1..r - 1$ we choose to either run $k$-means++ or the repeated $k$-means# (algorithm $A$ in the previous subsection), i.e., $(a_i, b_i) = (1, O(\log k))$ or $(3 \log k, O(1))$ for each $i$. For $i = r$, we choose $k$-means++, i.e., $(a_r, b_r) = (1, O(\log k))$ (we are interested in outputting exactly $k$ centers as the final solution). Let $q$ denote the number of

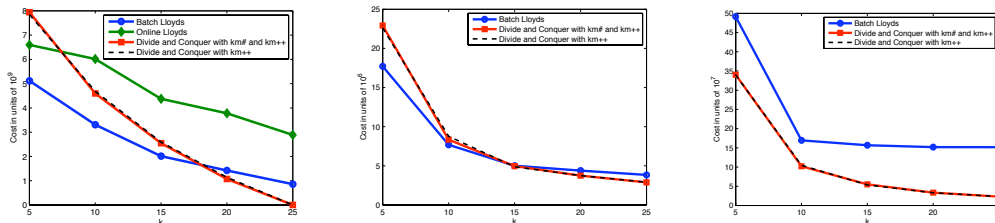

Figure 1: Cost vs. $k$: (a) Mixtures of gaussians simulation, (b) Cloud data, (c) Spam data,.

indexes $i \in [r-1]$ such that $(a_i, b_i) = (3\log k, O(1))$. By the above discussion, the memory is used optimally if $M = rn^{1/r}k(3\log k)^{q/r}$, in which case the final approximation guarantee will be $\tilde{c}^{r-1}(\log k)^{r-q}$, for some global $\tilde{c} > 0$. We concentrate on the case $M$ growing polynomially in $n$, say $M = n^{\alpha}$ for some $\alpha < 1$. In this case, the memory optimality constraint implies $r = 1/\alpha$ for $n$ large enough (regardless of the choice of $q$). This implies that the final approximation guarantee is best if $q = r - 1$, in other words, we choose the repeated $k$-means# for levels $1..r-1$, and $k$-means++ for level $r$. Summarizing, we get:

**Theorem 3.2.** *If there is access to memory of size $M = n^{\alpha}$ for some fixed $\alpha > 0$, then for sufficiently large $n$ the best application of the multi-level scheme described above is obtained by running $r = \lfloor \alpha \rfloor = \lfloor \log n / \log M \rfloor$ levels, and choosing the repeated $k$-means# for all but the last level, in which $k$-means++ is chosen. The resulting algorithm is a randomized one-pass streaming approximation to $k$-means, with an approximation ratio of $O(\tilde{c}^{r-1}(\log k))$, for some global $\tilde{c} > 0$. The running time of the algorithm is $O(dnk^2 \log n \log k)$.*

We should compare the above multi-level streaming algorithm with the state-of-art (in terms of memory vs. approximation tradeoff) streaming algorithm for the $k$-median problem. Charikar, Callaghan, and Panigrahy [CCP03] give a one-pass streaming algorithm for the $k$-median problem which gives a constant factor approximation and uses $O(k \cdot \text{poly} \log(n))$ memory. The main problem with this algorithm from a practical point of view is that the average processing time per item is large. It is proportional to the amount of memory used, which is poly-logarithmic in $n$. This might be undesirable in practical scenarios where we need to process a data item quickly when it arrives. In contrast, the average per item processing time using the divide-and-conquer-strategy is constant and furthermore the algorithm can be pipelined (*i.e.* data items can be temporarily stored in a memory buffer and quickly processed before the the next memory buffer is filled). So, even if [CCP03] can be extended to the $k$-means setting, streaming algorithms based on the divide-and-conquer-strategy would be more interesting from a practical point of view.

## 4 Experiments

**Datasets.** In our discussion, $n$ denotes the number of points in the data, $d$ denotes the dimension, and $k$ denotes the number of clusters. Our first evaluation, detailed in Tables 1a)-c) and Figure 1, compares our algorithms on the following data: (1) *norm25* is synthetic data generated in the following manner: we choose 25 random vertices from a 15 dimensional hypercube of side length 500. We then add 400 gaussian random points (with variance 1) around each of these points.[8] So, for this data $n = 10,000$ and $d = 15$. The optimum cost for $k = 25$ is $1.5026 \times 10^5$. (2) The UCI *Cloud* dataset consists of cloud cover data [AN07]. Here $n = 1024$ and $d = 10$. (3) The UCI *Spambase* dataset is data for an e-mail spam detection task [AN07]. Here $n = 4601$ and $d = 58$.

To compare against a baseline method known to be used in practice, we used Lloyd's algorithm, commonly referred to as the $k$-means algorithm. Standard Lloyd's algorithm operates in the batch setting, which is an easier problem than the one-pass streaming setting, so we ran experiments with this algorithm to form a baseline. We also compare to an online version of Lloyd's algorithm, however the performance is worse than the batch version, and our methods, for all problems, so we

| k | BL | OL | DC-1 | DC-2 | BL | OL | DC-1 | DC-2 |
|---|---|---|---|---|---|---|---|---|
| 5 | $5.1154 \cdot 10^9$ | $6.5967 \cdot 10^9$ | $7.9398 \cdot 10^9$ | $7.8474 \cdot 10^9$ | 1.25 | 1.32 | 14.37 | 9.93 |
| 10 | $3.3080 \cdot 10^9$ | $6.0146 \cdot 10^9$ | $4.5954 \cdot 10^9$ | $4.6829 \cdot 10^9$ | 2.05 | 2.45 | 45.39 | 21.09 |
| 15 | $2.0123 \cdot 10^9$ | $4.3743 \cdot 10^9$ | $2.5468 \cdot 10^9$ | $2.5898 \cdot 10^9$ | 3.88 | 3.49 | 95.22 | 30.34 |
| 20 | $1.4225 \cdot 10^9$ | $3.7794 \cdot 10^9$ | $1.0718 \cdot 10^9$ | $1.1403 \cdot 10^9$ | 8.62 | 4.69 | 190.73 | 41.49 |
| 25 | $0.8602 \cdot 10^9$ | $2.8859 \cdot 10^9$ | $2.7842 \cdot 10^5$ | $2.7298 \cdot 10^5$ | 13.13 | 6.04 | 283.19 | 53.07 |

| k | BL | OL | DC-1 | DC-2 | BL | OL | DC-1 | DC-2 |
|---|---|---|---|---|---|---|---|---|
| 5 | $1.7707 \cdot 10^7$ | $1.2401 \cdot 10^8$ | $2.2924 \cdot 10^7$ | $2.2617 \cdot 10^7$ | 1.12 | 0.13 | 1.73 | 0.92 |
| 10 | $0.7683 \cdot 10^7$ | $8.5684 \cdot 10^7$ | $8.3363 \cdot 10^6$ | $8.7788 \cdot 10^6$ | 1.20 | 0.25 | 5.64 | 1.87 |
| 15 | $0.5012 \cdot 10^7$ | $8.4633 \cdot 10^7$ | $4.9667 \cdot 10^6$ | $4.8806 \cdot 10^6$ | 2.18 | 0.35 | 10.98 | 2.67 |
| 20 | $0.4388 \cdot 10^7$ | $6.5110 \cdot 10^7$ | $3.7479 \cdot 10^6$ | $3.7536 \cdot 10^6$ | 2.59 | 0.47 | 25.72 | 4.19 |
| 25 | $0.3839 \cdot 10^7$ | $6.3758 \cdot 10^7$ | $2.8895 \cdot 10^6$ | $2.9014 \cdot 10^6$ | 2.43 | 0.52 | 36.17 | 4.82 |

| k | BL | OL | DC-1 | DC-2 | BL | OL | DC-1 | DC-2 |
|---|---|---|---|---|---|---|---|---|
| 5 | $4.9139 \cdot 10^8$ | $1.7001 \cdot 10^9$ | $3.4021 \cdot 10^8$ | $3.3963 \cdot 10^8$ | 9.68 | 0.70 | 11.65 | 5.14 |
| 10 | $1.6952 \cdot 10^8$ | $1.6930 \cdot 10^9$ | $1.0206 \cdot 10^8$ | $1.0463 \cdot 10^8$ | 34.78 | 1.31 | 40.14 | 9.75 |
| 15 | $1.5670 \cdot 10^8$ | $1.4762 \cdot 10^9$ | $5.5095 \cdot 10^7$ | $5.3557 \cdot 10^7$ | 67.54 | 1.88 | 77.75 | 14.41 |
| 20 | $1.5196 \cdot 10^8$ | $1.4766 \cdot 10^9$ | $3.3400 \cdot 10^7$ | $3.2994 \cdot 10^7$ | 100.44 | 2.57 | 194.01 | 22.76 |
| 25 | $1.5168 \cdot 10^8$ | $1.4754 \cdot 10^9$ | $2.3151 \cdot 10^7$ | $2.3391 \cdot 10^7$ | 109.41 | 3.04 | 274.42 | 27.10 |

Table 1: Columns 2-5 have the clustering cost and columns 6-9 have time in sec. a) norm25 dataset, b) Cloud dataset, c) Spambase dataset.

| Memory/#levels | Cost | Time | Memory/#levels | Cost | Time | Memory/#levels | Cost | Time |
|---|---|---|---|---|---|---|---|---|
| 1024/0 | $8.74 \cdot 10^6$ | 5.5 | 2048/0 | $5.78 \cdot 10^4$ | 30 | 4601/0 | $1.06 \cdot 10^8$ | 34 |
| 480/1 | $8.59 \cdot 10^6$ | 3.6 | 1250/1 | $5.36 \cdot 10^4$ | 25 | 880/1 | $0.99 \cdot 10^8$ | 20 |
| 360/2 | $8.61 \cdot 10^6$ | 3.8 | 1125/2 | $5.15 \cdot 10^4$ | 26 | 600/2 | $1.03 \cdot 10^8$ | 19.5 |

Table 2: Multi-level hierarchy evaluation: a) Cloud dataset, $k = 10$, b) A subset of norm25 dataset, $n = 2048, k = 25$, c) Spambase dataset, $k = 10$. The memory size decreases as the number of levels of the hierarchy increases. (0 levels means running batch k-means++ on the data.)

do not include it in our plots for the real data sets.[9] Tables 1a)-c) shows average $k$-means cost (over 10 random restarts for the randomized algorithms: all but Online Lloyd's) for these algorithms:
(1) **BL**: Batch Lloyd's, initialized with random centers in the input data, and run to convergence.[10]
(2) **OL**: Online Lloyd's.
(3) **DC-1**: The simple 1-stage divide and conquer algorithm of Section 3.2.
(4) **DC-2**: The simple 1-stage divide and conquer algorithm 3 of Section 3.1. The sub-algorithms used are A = "run k-means++ $3 \cdot \log n$ times and pick best clustering," and A' is k-means++. In our context, k-means++ and k-means# are only the seeding step, not followed by Lloyd's algorithm.

In all problems, our streaming methods achieve much lower cost than Online Lloyd's, for all settings of $k$, and lower cost than Batch Lloyd's for most settings of $k$ (including the correct $k = 25$, in norm25). The gains with respect to batch are noteworthy, since the batch problem is less constrained than the one-pass streaming problem. The performance of DC-1 and DC-2 is comparable.

Table 2 shows an evaluation of the one-pass multi-level hierarchical algorithm of Section 3.3, on the different datasets, simulating different memory restrictions. Although our worst-case theoretical results imply an exponential clustering cost as a function of the number of levels, our results show a far more optimistic outcome in which adding levels (and limiting memory) actually improves the outcome. We conjecture that our data contains enough information for clustering even on chunks that fit in small buffers, and therefore the results may reflect the benefit of the hierarchical implementation.

**Acknowledgements.** We thank Sanjoy Dasgupta for suggesting the study of approximation algorithms for $k$-means in the streaming setting, for excellent lecture notes, and for helpful discussions.

## Footnotes

[3]For a comprehensive survey of streaming results and literature, refer to [M05].

[4]In recent, independent work, Aggarwal, Deshpande, and Kannan [ADK09] extend the seeding procedure of $k$-means++ to obtain a constant factor approximation algorithm which outputs $O(k)$ centers. They use similar techniques to ours, but reduce the number of centers by using a stronger concentration property.

[5]It may be interesting future work to analyze our algorithm in special cases, such as well-separated clusters.

[6]For the unweighted case, we can assume that $w(x) = 1$ for all $x$.

[7]We assume all quotients are integers for simplicity of the proof, but note that fractional blocks would arise in practice.

[8]Testing clustering algorithms on this simulation distribution was inspired by [AV07].

[9]Despite the poor performance we observed, this algorithm is apparently used in practice, see [Das08].

[10]We measured convergence by change in cost less than 1.

# References

[ADK09]  Ankit Aggarwal, Amit Deshpande and Ravi Kannan: Adaptive Sampling for k-means Clustering. APPROX, 2009.

[AL09]  Nir Ailon and Edo Liberty: Correlation Clustering Revisited: The "True" Cost of Error Minimization Problems. To appear in ICALP 2009.

[AMS96]  Noga Alon, Yossi Matias, and Mario Szegedy.: The space complexity of approximating the frequency moments. In Proceedings of the Twenty-Eighth Annual ACM Symposium on Theory of Computing, pages 20–29, 1996.

[AV06]  David Arthur and Sergei Vassilvitskii: Worst-case and smoothed analyses of the icp algorithm, with an application to the k-means method. FOCS, 2006

[AV07]  David Arthur and Sergei Vassilvitskii: $k$-means++: the advantages of careful seeding. SODA, 2007.

[AGKM+04]  V. Arya, N. Garg, R. Khandekar, A. Meyerson, K. Munagala, and V. Pandit: Local search heuristics for k-median and facility location problems. Siam Journal of Computing, 33(3):544–562, 2004.

[AN07]  A. Asuncion and D.J. Newman: UCI Machine Learning Repository. *http://www.ics.uci.edu/~mlearn/MLRepository.html*, University of California, Irvine, School of Information and Computer Sciences, 2007.

[BBG09]  Maria-Florina Balcan, Avrim Blum, and Anupam Gupta: Approximate Clustering without the Approximation. SODA, 2009.

[BL08]  S. Ben-David and U. von Luxburg: Relating clustering stability to properties of cluster boundaries. COLT, 2008

[CCP03]  Moses Charikar and Liadan O'Callaghan and Rina Panigrahy: Better streaming algorithms for clustering problems. STOC, 2003.

[CG99]  Moses Charikar and Sudipto Guha: Improved combinatorial algorithms for the facility location and $k$-medians problem. FOCS, 1999.

[CMTS02]  M. Charikar, S. Guha , E Tardos, and D. Shmoys: A Constant Factor Approximation Algorithm for the k-Median Problem. Journal of Computer and System Sciences, 2002.

[CR08]  Kamalika Chaudhuri and Satish Rao: Learning Mixtures of Product Distributions using Correlations and Independence. COLT, 2008.

[Das08]  Sanjoy Dasgupta.: Course notes, CSE 291: Topics in unsupervised learning. *http://www-cse.ucsd.edu/ dasgupta/291/index.html*, University of California, San Diego, Spring 2008.

[Gon85]  T. F. Gonzalez: Clustering to minimize the maximum intercluster distance. Theoretical Computer Science, 38, pages 293–306, 1985.

[GMMM+03]  Sudipto Guha, Adam Meyerson, Nina Mishra, Rajeev Motwani, and Liadan O'Callaghan: Clustering Data Streams: Theory and Practice. IEEE Transactions on Knowledge and Data Engineering, 15(3): 515–528, 2003.

[Ind99]  Piotr Indyk: Sublinear Time Algorithms for Metric Space Problems. STOC, 1999.

[JV01]  K. Jain and Vijay Vazirani: Approximation Algorithms for Metric Facility Location and k-Median Problems Using the Primal-Dual Schema and Lagrangian Relaxation. Journal of the ACM. 2001.

[KMNP+04]  T. Kanungo, D. M. Mount, N. Netanyahu, C. Piatko, R. Silverman, and A. Y. Wu: A Local Search Approximation Algorithm for k-Means Clustering, Computational Geometry: Theory and Applications, 28, 89-112, 2004.

[LV92]  J. Lin and J. S. Vitter: Approximation Algorithms for Geometric Median Problems. Information Processing Letters, 1992.

[McG07]  Andrew McGregor: Processing Data Streams. Ph.D. Thesis, Computer and Information Science, University of Pennsylvania, 2007.

[M05]  S. Muthukrishnan: Data Streams: Algorithms and Applications, NOW Publishers, Inc., Hanover MA

[ORSS06]  Rafail Ostrovsky, Yuval Rabani, Leonard J. Schulman, Chaitanya Swamy: The effectiveness of Lloyd-type methods for the k-means problem. FOCS, 2006.

[VW02]  V. Vempala and G. Wang: A spectral algorithm of learning mixtures of distributions. pages 113–123, FOCS, 2002.

